# Tangent Prop – A formalism for specifying selected invariances in an adaptive network

**Patrice Simard**
AT&T Bell Laboratories
101 Crawford Corner Rd
Holmdel, NJ 07733

**Bernard Victorri**
Université de Caen
Caen 14032 Cedex
France

**Yann Le Cun**
AT&T Bell Laboratories
101 Crawford Corner Rd
Holmdel, NJ 07733

**John Denker**
AT&T Bell Laboratories
101 Crawford Corner Rd
Holmdel, NJ 07733

## Abstract

In many machine learning applications, one has access, not only to training data, but also to some high-level *a priori* knowledge about the desired behavior of the system. For example, it is known in advance that the output of a character recognizer should be invariant with respect to small spatial distortions of the input images (translations, rotations, scale changes, etcetera).

We have implemented a scheme that allows a network to learn the derivative of its outputs with respect to distortion operators of our choosing. This not only reduces the learning time and the amount of training data, but also provides a powerful *language* for specifying what generalizations we wish the network to perform.

## 1 INTRODUCTION

In machine learning, one very often knows more about the function to be learned than just the training data. An interesting case is when certain *directional derivatives* of the desired function are known at certain points. For example, an image

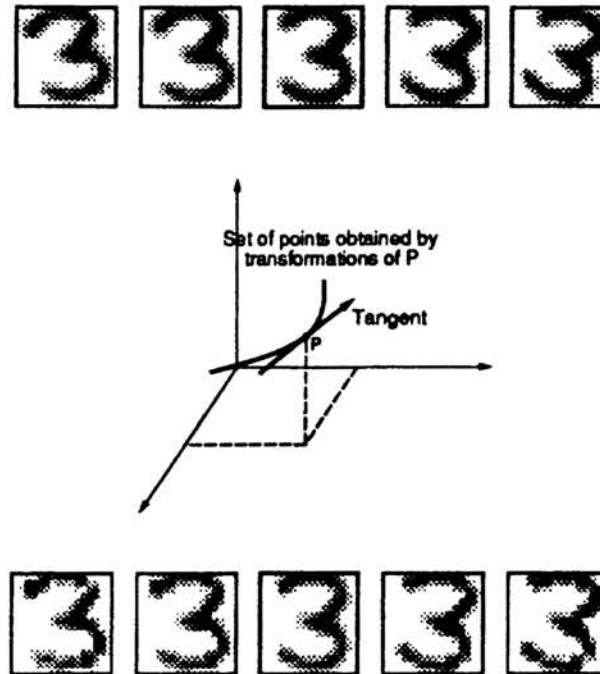

Figure 1: Top: Small rotations of an original digital image of the digit "3" (center). Middle: Representation of the effect of the rotation in the input vector space space (assuming there are only 3 pixels). Bottom: Images obtained by moving along the tangent to the transformation curve for the same original digital image (middle).

recognition system might need to be invariant with respect to small distortions of the input image such as translations, rotations, scalings, etc.; a speech recognition system might need to be invariant to time distortions or pitch shifts. In other words, the derivative of the system's output should be equal to zero when the input is transformed in certain ways.

Given a large amount of training data and unlimited training time, the system could learn these invariances from the data alone, but this is often infeasible. The limitation on data can be overcome by training the system with additional data obtained by distorting (translating, rotating, etc.) the original patterns (Baird, 1990). The top of Fig. 1 shows artificial data generated by rotating a digital image of the digit "3" (with the original in the center). This procedure, called the "distortion model", has two drawbacks. First, the user must choose the magnitude of distortion and how many instances should be generated. Second, and more importantly, the distorted data is highly correlated with the original data. This makes traditional learning algorithms such as backpropagation very inefficient. The distorted data carries only a very small incremental amount of information, since the distorted patterns are not very different from the original ones. It may not be possible to adjust the learning system so that learning the invariances proceeds at a reasonable rate while learning the original points is non-divergent.

The key idea in this paper is that it is possible to directly learn the effect on the output of distorting the input, *independently* from learning the undistorted

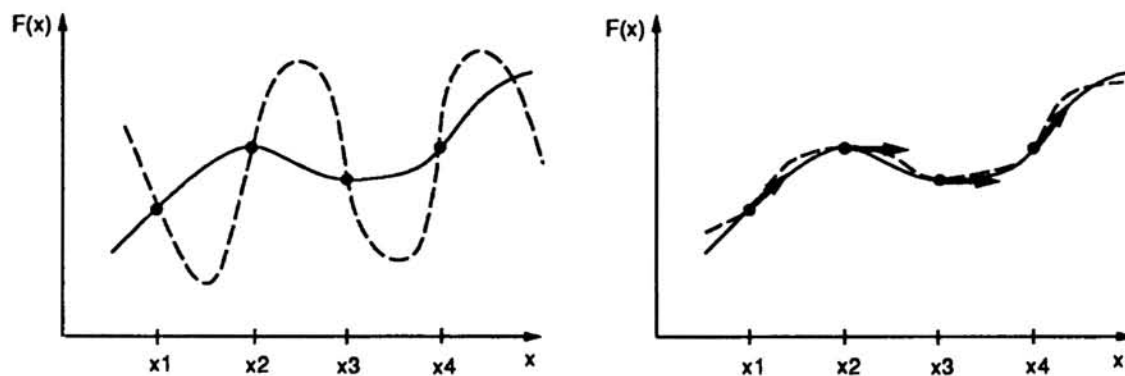

Figure 2: Learning a given function (solid line) from a limited set of example ($x_1$ to $x_4$). The fitted curves are shown in dotted line. Top: The only constraint is that the fitted curve goes through the examples. Bottom: The fitted curves not only goes through each examples but also its derivatives evaluated at the examples agree with the derivatives of the given function.

patterns. When a pattern $P$ is transformed (e.g. rotated) with a transformation $s$ that depends on one parameter $\alpha$ (e.g. the angle of the rotation), the set of all the transformed patterns $S(P) = \{s(\alpha, P) \; \forall \alpha\}$ is a one dimensional curve in the vector space of the inputs (see Fig. 1). In certain cases, such as rotations of digital images, this curve must be made continuous using smoothing techniques, as will be shown below. When the set of transformations is parameterized by $n$ parameters $\alpha_i$ (rotation, translation, scaling, etc.), $S(P)$ is a manifold of at most $n$ dimensions. The patterns in $S(P)$ that are obtained through *small* transformations of $P$, i.e. the part of $S(P)$ that is close to $P$, can be approximated by a plane tangent to the manifold $S(P)$ at point $P$. Small transformations of $P$ can be obtained by adding to $P$ a linear combination of vectors that span the tangent plane (tangent vectors). The images at the bottom of Fig. 1 were obtained by that procedure. More importantly, the tangent vectors can be used to specify high order constraints on the function to be learned, as explained below.

To illustrate the method, consider the problem of learning a single-valued function $F$ from a limited set of examples. Fig. 2 (left) represents a simple case where the desired function $F$ (solid line) is to be approximated by a function $G$ (dotted line) from four examples $\{(x_i, F(x_i))\}_{i=1,2,3,4}$. As exemplified in the picture, the fitted function $G$ largely disagrees with the desired function $F$ between the examples. If the functions $F$ and $G$ are assumed to be differentiable (which is generally the case), the approximation $G$ can be greatly improved by requiring that $G$'s derivatives evaluated at the points $\{x_i\}$ are equal to the derivatives of $F$ at the same points (Fig. 2 right). This result can be extended to multidimensional inputs. In this case, we can impose the equality of the derivatives of $F$ and $G$ in *certain directions*, not necessarily in all directions of the input space.

Such constraints find immediate use in traditional learning problems. It is often the case that *a priori* knowledge is available on how the desired function varies with

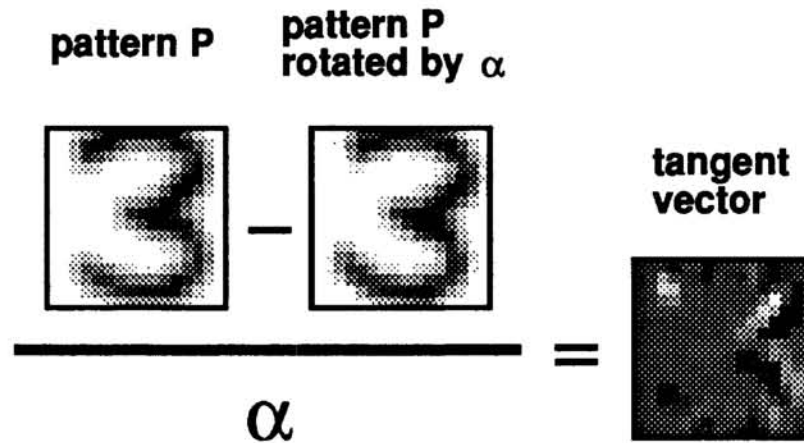

Figure 3: How to compute a tangent vector for a given transformation (in this case a rotation).

respect to some transformations of the input. It is straightforward to derive the corresponding constraint on the directional derivatives of the fitted function $G$ in the directions of the transformations (previously named tangent vectors). Typical examples can be found in pattern recognition where the desired classification function is known to be invariant with respect to some transformation of the input such as translation, rotation, scaling, etc., in other words, the directional derivatives of the classification function in the directions of these transformations is zero.

## 2    IMPLEMENTATION

The implementation can be divided into two parts. The first part consists in computing the tangent vectors. This part is independent from the learning algorithm used subsequently. The second part consists in modifying the learning algorithm (for instance backprop) to incorporate the information about the tangent vectors.

**Part I:** Let $x$ be an input pattern and $s$ be a transformation operator acting on the input space and depending on a parameter $\alpha$. If $s$ is a rotation operator for instance, then $s(\alpha, x)$ denotes the input $x$ rotated by the angle $\alpha$. We will require that the transformation operator $s$ be differentiable with respect to $\alpha$ and $x$, and that $s(0, x) = x$. The tangent vector is by definition $\partial s(\alpha, x)/\partial \alpha$. It can be approximated by a finite difference, as shown in Fig. 3. In the figure, the input space is a 16 by 16 pixel image and the patterns are images of handwritten digits. The transformations considered are rotations of the digit images. The tangent vector is obtained in two steps. First the image is rotated by an infinitesimal amount $\alpha$. This is done by computing the rotated coordinates of each pixel and interpolating the gray level values at the new coordinates. This operation can be advantageously combined with some smoothing using a convolution. A convolution with a Gaussian provides an efficient interpolation scheme in $O(nm)$ multiply-adds, where $n$ and $m$ are the (gaussian) kernel and image sizes respectively. The next step is to subtract (pixel by pixel) the rotated image from the original image and to divide the result

by the scalar $\alpha$ (see Fig. 3). If $k$ types of transformations are considered, there will be $k$ different tangent vectors per pattern. For most algorithms, these do not require any storage space since they can be generated as needed from the original pattern at negligible cost.

**Part II:** Tangent prop is an extension of the backpropagation algorithm, allowing it to learn directional derivatives. Other algorithms such as radial basis functions can be extended in a similar fashion.

To implement our idea, we will modify the usual weight-update rule:

$$\Delta w = -\eta \frac{\partial E}{\partial w} \quad \text{is replaced with} \quad \Delta w = -\eta \frac{\partial}{\partial w}(E + \mu E_r) \qquad (1)$$

where $\eta$ is the learning rate, $E$ the usual objective function, $E_r$ an additional objective function (a *regularizer*) that measures the discrepancy between the actual and desired directional derivatives in the directions of some selected transformations, and $\mu$ is a weighting coefficient.

Let $x$ be an input pattern, $y = G(x)$ be the input-output function of the network. The regularizer $E_r$ is of the form

$$E_r = \sum_{x \in \text{trainingset}} E_r(x)$$

where $E_r(x)$ is

$$E_r(x) = \sum_i \left\| K_i(x) - \left( \frac{\partial G(s_i(\alpha, x))}{\partial \alpha} \right)_{\alpha=0} \right\|^2 \qquad (2)$$

Here, $K_i(x)$ is the desired directional derivative of $G$ in the direction induced by transformation $s_i$ applied to pattern $x$. The second term in the norm symbol is the actual directional derivative, which can be rewritten as

$$\frac{\partial G(s_i(\alpha, x))}{\partial \alpha}\bigg|_{\alpha=0} = G'(x) \cdot \frac{\partial s_i(\alpha, x)}{\partial \alpha}\bigg|_{\alpha=0}$$

where $G'(x)$ is the Jacobian of $G$ for pattern $x$, and $\partial s_i(\alpha, x)/\partial \alpha$ is the *tangent vector* associated to transformation $s_i$ as described in Part I. Multiplying the tangent vector by the Jacobian involves one forward propagation through a "linearized" version of the network. In the special case where *local invariance* with respect to the $s_i$'s is desired, $K_i(x)$ is simply set to 0.

**Composition of transformations:** The theory of Lie groups (Gilmore, 1974) ensures that compositions of local (small) transformations $s_i$ correspond to linear combinations of the corresponding tangent vectors (the local transformations $s_i$ have a structure of Lie algebra). Consequently, if $E_r(x) = 0$ is verified, the network derivative in the direction of a linear combination of the tangent vectors is equal to the same linear combination of the desired derivatives. In other words if the network is successfully trained to be locally invariant with respect to, say, horizontal translation and vertical translations, it will be invariant with respect to compositions thereof.

We have derived and implemented an efficient algorithm, "tangent prop", for performing the weight update (Eq. 1). It is analogous to ordinary backpropagation,

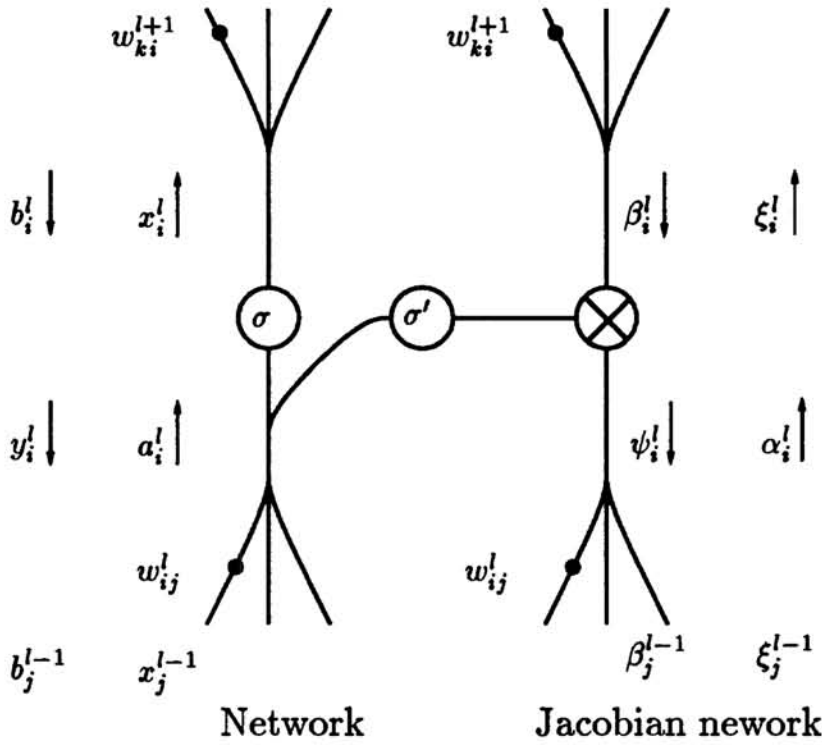

Figure 4: forward propagated variables $(a, x, \alpha, \xi)$, and backward propagated variables $(b, y, \beta, \psi)$ in the regular network (roman symbols) and the Jacobian (linearized) network (greek symbols)

but in addition to propagating neuron activations, it also propagates the tangent vectors. The equations can be easily derived from Fig. 4.

Forward propagation:

$$a_i^l = \sum_j w_{ij}^l x_j^{l-1} \qquad x_i^l = \sigma(a_i^l) \tag{3}$$

Tangent forward propagation:

$$\alpha_i^l = \sum_j w_{ij}^l \xi_j^{l-1} \qquad \xi_i^l = \sigma'(a_i^l)\alpha_i^l \tag{4}$$

Tangent gradient backpropagation:

$$\beta_i^l = \sum_k w_{ki}^{l+1} \psi_k^{l+1} \qquad \psi_i^l = \sigma'(a_i^l)\beta_i^l \tag{5}$$

Gradient backpropagation:

$$b_i^l = \sum_k w_{ki}^{l+1} y_k^{l+1} \qquad y_i^l = \sigma'(a_i^l)b_i^l + \sigma''(a_i^l)\alpha_i^l\beta_i \tag{6}$$

Weight update:

$$\frac{\partial[E(W, U_p) + \mu E_r(W, U_p, T_p)]}{\partial w_{ij}^l} = x_j^{l-1} y_i^l + \mu \xi_j^{l-1} \psi_i^l \tag{7}$$

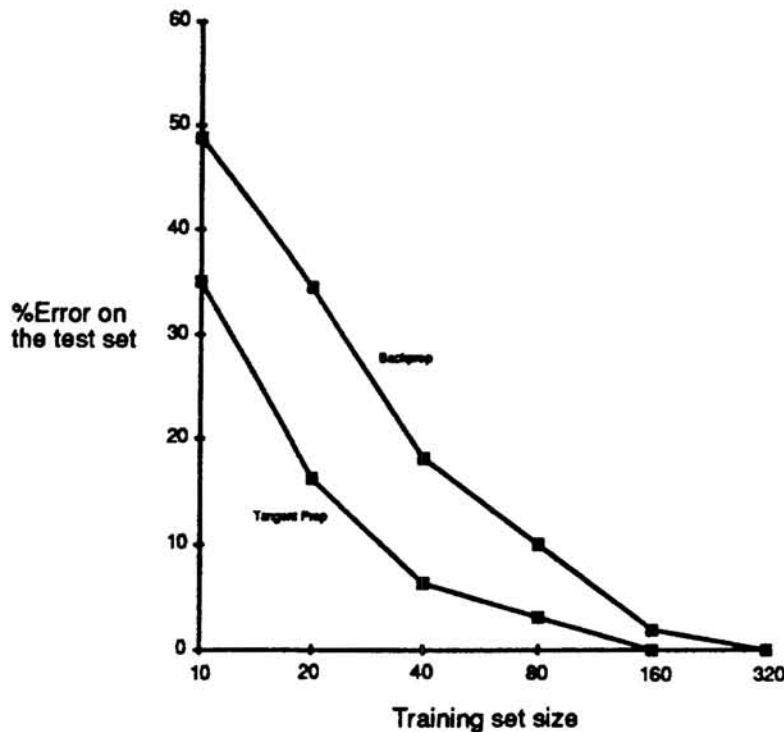

Figure 5: Generalization performance curve as a function of the training set size for the tangent prop and the backprop algorithms

The regularization parameter $\mu$ is tremendously important, because it determines the tradeoff between minimizing the usual objective function and minimizing the directional derivative error.

## 3   RESULTS

Two experiments illustrate the advantages of tangent prop. The first experiment is a classification task, using a small (linearly separable) set of 480 binarized hand-written digit. The training sets consist of 10, 20, 40, 80, 160 or 320 patterns, and the training set contains the remaining 160 patterns. The patterns are smoothed using a gaussian kernel with standard deviation of one half pixel. For each of the training set patterns, the tangent vectors for horizontal and vertical translation are computed. The network has two hidden layers with locally connected shared weights, and one output layer with 10 units (5194 connections, 1060 free parameters) (Le Cun, 1989). The generalization performance as a function of the training set size for traditional backprop and tangent prop are compared in Fig. 5. We have conducted additional experiments in which we implemented not only translations but also rotations, expansions and hyperbolic deformations. This set of 6 generators is a basis for all linear transformations of coordinates for two dimensional images. It is straightforward to implement other generators including gray-level-shifting, "smooth" segmentation, local continuous coordinate transformations and independent image segment transformations.

The next experiment is designed to show that in applications where data is highly

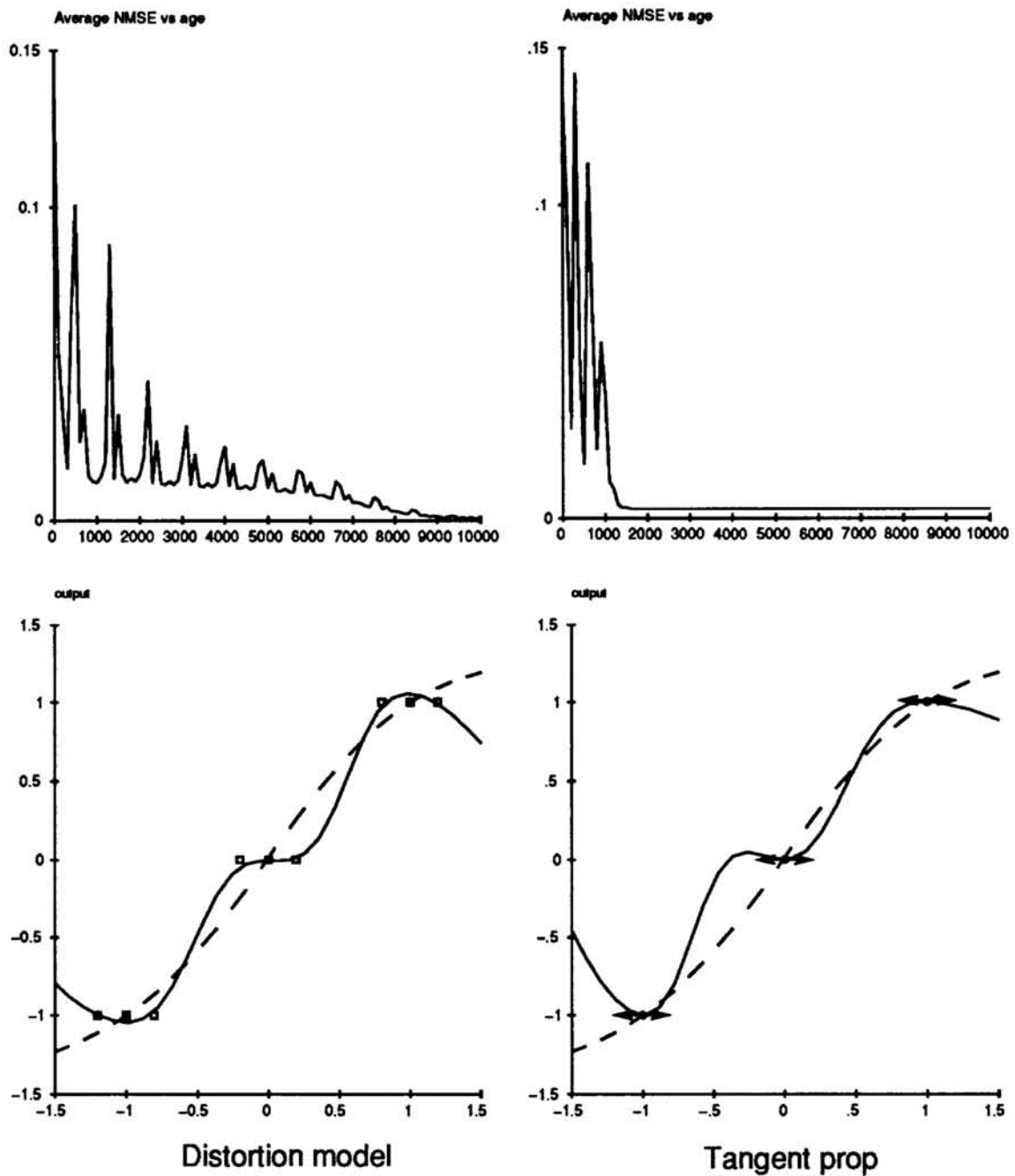

Figure 6: Comparison of the distortion model (left column) and tangent prop (right column). The top row gives the learning curves (error versus number of sweeps through the training set). The bottom row gives the final input-output function of the network; the dashed line is the result for unadorned back prop.

correlated, tangent prop yields a large speed advantage. Since the distortion model implies adding lots of highly correlated data, the advantage of tangent prop over the distortion model becomes clear.

The task is to approximate a function that has plateaus at three locations. We want to enforce local invariance near each of the training points (Fig. 6, bottom). The network has one input unit, 20 hidden units and one output unit. Two strategies are possible: either generate a small set of training point covering each of the plateaus (open squares on Fig. 6 bottom), or generate one training point for each plateau (closed squares), and enforce local invariance around them (by setting the desired derivative to 0). The training set of the former method is used as a measure the performance for both methods. All parameters were adjusted for approximately optimal performance in all cases. The learning curves for both models are shown in Fig. 6 (top). Each sweep through the training set for tangent prop is a little faster since it requires only 6 forward propagations, while it requires 9 in the distortion model. As can be seen, stable performance is achieved after 1300 sweeps for the tangent prop, versus 8000 for the distortion model. The overall speedup is therefore about 10.

Tangent prop in this example can take advantage of a very large regularization term. The distortion model is at a disadvantage because the only parameter that effectively controls the amount of regularization is the magnitude of the distortions, and this cannot be increased to large values because the right answer is only invariant under *small* distortions.

## 4    CONCLUSIONS

When *a priori* information about invariances exists, this information must be made available to the adaptive system. There are several ways of doing this, including the distortion model and tangent prop. The latter may be much more efficient in some applications, and it permits separate control of the emphasis and learning rate for the invariances, relative to the original training data points. Training a system to have zero derivatives in some directions is a powerful tool to express invariances to transformations of our choosing. Tests of this procedure on large-scale applications (handwritten zipcode recognition) are in progress.

## References

Baird, H. S. (1990). Document Image Defect Models. In *IAPR 1990 Workshop on Sytactic and Structural Pattern Recognition*, pages 38–46, Murray Hill, NJ.

Gilmore, R. (1974). *Lie Groups, Lie Algebras and some of their Applications*. Wiley, New York.

Le Cun, Y. (1989). Generalization and Network Design Strategies. In Pfeifer, R., Schreter, Z., Fogelman, F., and Steels, L., editors, *Connectionism in Perspective*, Zurich, Switzerland. Elsevier. an extended version was published as a technical report of the University of Toronto.
